# Infinite State Bayesian Networks

**Max Welling,**[*] **Ian Porteous, Evgeniy Bart**[†]
Donald Bren School of Information and Computer Sciences
University of California Irvine
Irvine, CA 92697-3425 USA
{welling,iporteou}@ics.uci.edu, bart@caltech.edu

## Abstract

A general modeling framework is proposed that unifies nonparametric-Bayesian models, topic-models and Bayesian networks. This class of infinite state Bayes nets (ISBN) can be viewed as directed networks of 'hierarchical Dirichlet processes' (HDPs) where the domain of the variables can be structured (e.g. words in documents or features in images). We show that collapsed Gibbs sampling can be done efficiently in these models by leveraging the structure of the Bayes net and using the forward-filtering-backward-sampling algorithm for junction trees. Existing models, such as nested-DP, Pachinko allocation, mixed membership stochastic block models as well as a number of new models are described as ISBNs. Two experiments have been performed to illustrate these ideas.

## 1   Introduction

Bayesian networks remain the cornerstone of modern AI. They have been applied to a wide range of problems both in academia as well as in industry. A recent development in this area is a class of Bayes nets known as topic models (e.g. LDA [1]) which are well suited for structured data such as text or images. A recent statistical sophistication of topic models is a nonparametric extension known as HDP [2], which adaptively infers the number of topics based on the available data.

This paper has the goal of bridging the gap between these three developments. We propose a general modeling paradigm, the "infinite state Bayes net" (ISBN), that incorporates these three aspects. We consider models where the variables may have the nested structure of documents and images, may have infinite discrete state spaces, and where the random variables are related through the intuitive causal dependencies of a Bayes net. ISBN's can be viewed as collections of HDP "modules" connected together to form a network. Inference in these networks is achieved through a two-stage Gibbs sampler, which combines the "forward-filtering-backward-sampling algorithm" [3] extended to junction trees and the direct assignment sampler for HDPs [2].

## 2   Bayes Net Structure for ISBN

Consider observed random variables $\mathbf{x}^A \triangleq \{\mathbf{x}^a\}$, $a = 1..A$. These variables can take values in an arbitrary domain. In the following we will assume that $\mathbf{x}^a$ is sampled from a (conditional) distribution in the exponential family. We will also introduce hidden (unobserved, latent) variables $\{\mathbf{z}^b\}$, $b = 1..B$ which will always take discrete values. The indices $a, b$ thus index the nodes of the Bayesian network.

We will introduce a separate index, e.g. $\mathfrak{n}^a$, to label observations. In the simplest setting we assume IID data $\mathfrak{n} = i$, i.e. $N$ independent identically distributed observations for each variable. We will

---

[*] On sabbatical at Radboud University Nijmegen, Netherlands, Dept. of Biophysics.

[†] Joint appointment at California Institute of Technology, USA, Dept. of Electrical Engineering.

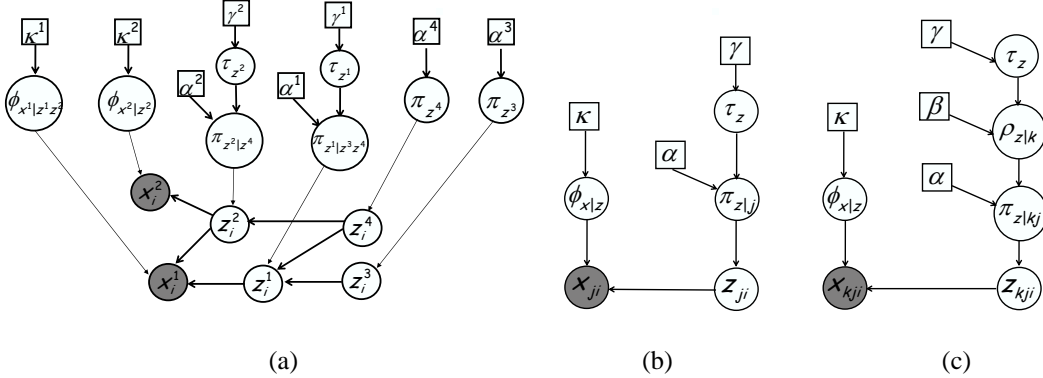

Figure 1: Graphical representation of (a) Unstructured infinite state Bayesian network, (b) HDP, (c) H2DP.

however also be interested in more structured data, such as words in documents, where the index $\mathfrak{n}$ can be decomposed into e.g. $\mathfrak{n} = (j, i_j)$. In this notation we think of $j$ as labelling a document and $i_j$ as labelling a word in document $j$. To simplify notation we will often write $\mathfrak{n} = (ji)$. It is straightforward to generalize to deeper nestings of indices, e.g. $\mathfrak{n} = (k, j_k, i_{j_k}) = (kji)$ where $k$ can index e.g. books, $j$ chapters and $i$ words. We interpret this as the *observed* structure in the data, as opposed to the latent structure which we seek to infer. The unobserved structure is labelled with the discrete "assignment variables" $\mathbf{z}^a_{\mathfrak{n}}$ which assign the object indexed by $\mathfrak{n}$ to latent groups (a.k.a. topics, factors, components).

The assignment variables $\mathbf{z}$ together with the observed variables $\mathbf{x}$ are organized into a Bayes net, where dependencies are encoded by the usual "conditional probability tables" (CPT), which we denote with $\phi^a_{x^a|\wp^a}$ for observed variables and $\boldsymbol{\pi}^b_{z^b|\wp^b}$ for latent variables[1]. Here, $\wp^a$ denotes the joint state of all the parent variables of $\mathbf{x}^a$ or $\mathbf{z}^b$. When a vertical bar is present we normalize over the variables to the left of it, e.g. $\sum_{x^a} \phi^a_{x^a|\wp^a} = 1$, $\forall a, \wp^a$. Note that CPTs are considered random variables and may themselves be indexed by (a subset of) $\mathfrak{n}$, e.g. $\phi_{x^a|\wp^a j}$.

We assume that each $\boldsymbol{\pi}^b$ is sampled from a Dirichlet prior: e.g. $\boldsymbol{\pi}_{z^b|\wp^b} \sim \mathcal{D}[\alpha^b \boldsymbol{\tau}_{z^b}]$ independently and identically for all values of $\wp^b$. The distribution $\boldsymbol{\tau}$ itself is Dirichlet distributed, $\boldsymbol{\tau}_{z^a} \sim \mathcal{D}[\gamma^a/K^a]$, where $K^a$ is the number of states for variable $\mathbf{z}^a$. We can put gamma priors on $\alpha^a, \gamma^a$ and consider them as random variables as well, but to keep things simple we will consider them fixed variables here. We refer to [4] for algorithms to learn them from data and to [5] and [2] for ways to infer them through sampling. In section 5 we further discuss these hierarchical priors.

In drawing BNs we will not include the plates to avoid cluttering the figures. However, it is always possible to infer the number of times variables in the BN are replicated by looking at its indices. For instance, the variable node labelled with $\boldsymbol{\pi}_{z^1|z^2 j}$ in Fig.3a stands for $K^{(2)} \times J$ IID copies of $\boldsymbol{\pi}^1$ sampled from $\boldsymbol{\tau}^1$.

## 3 Networks of HDPs

In Fig.1b we have drawn the finite version of the HDP. Here $\phi$ is a distribution over words, one for each topic value $\mathbf{z}$, and is often referred to a "topic distribution". Topic values are generated from a document specific distribution $\boldsymbol{\pi}$ which in turn are generated from a "mother distribution" over topics $\boldsymbol{\tau}$. As was shown in [2] one can take the infinite limit $K \to \infty$ in this model and arrive at the HDP. We will return to this infinite limit when we describe Gibbs sampling. In the following we will use the same graphical model for finite and infinite versions of ISBNs.

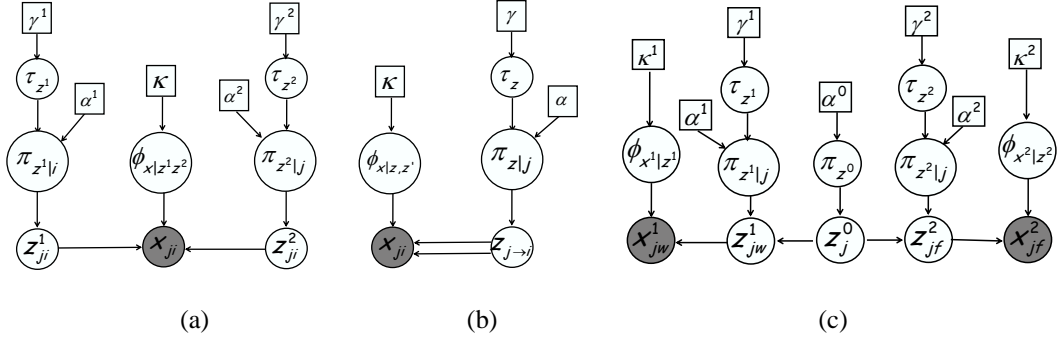

Figure 2: Graphical representation for (a) BiHDP, (b) Mixed membership stochastic block model and (c) the "multimedia" model.

One of the key features of the HDP is that topics are shared across all documents indexed by $j$. The reason for this is the distribution $\tau$: new states are "invented" at this level and become available to all other documents. In other words, there is a single state space for all copies of $\pi$. One can interpret $j$ is an instantiation of a dummy, fully observed random variable $\iota$. We could add this node to the BN as a parent of $\mathbf{z}$ (since $\pi$ depends on it) and reinterpret the statement of sharing topics as a fully connected transition matrix between states of $\iota$ to states of $\mathbf{z}$. This idea can be extended to a combination of fully observed parent variables and multiple unobserved parent variables, e.g. $\iota \rightarrow \mathbf{z}^2, \mathbf{z}^3, \iota$. Moreover, the child variables do not have to be observed either, so we can also replace $\mathbf{x} \rightarrow \mathbf{z}$. In this fashion we can connect together multiple vertical stacks $\tau \rightarrow \phi \rightarrow \mathbf{z}$ where each such module is part of a "virtual-HDP" where the joint child states act as virtual data and the joint parent states act as virtual document labels. Examples are given in Figs. 1a (infinite extension of a Bayes net with IID data items) and 3a (infinite extension of Pachinko Allocation).

## 4 Inference

To simplify the presentation we will now restrict attention to a Bayesian network where all CPTs are shared across all data-items (see Fig.1a). In this case data is unstructured, assumed IID and indexed by a flat index $\mathfrak{n} = i$. Instead of going through the detailed derivation, which is an extension of the derivation in [2] for HDP, we will describe the sampling process in the following.

There is a considerable body of empirical evidence which confirms that marginalizing out the variables $\pi, \phi$ will result in improved inference (e.g. [6, 7]). In this collapsed space, we sample two sets of variables alternatingly, $\{\mathbf{z}\}$ on the hand and $\{\tau\}$ on the other. First, we focus on the latter given $\mathbf{z}$ and notice that all $\tau$ are conditionally independent given $\mathbf{z}, \mathbf{x}$.

**Sampling** $\tau|(\mathbf{z}, \mathbf{x})$: Given $\mathbf{x}, \mathbf{z}$ we can compute count matrices[2] $N_{\mathbf{z}^b|\wp^b}$ and $N_{\mathbf{x}^a|\wp^a}$ as $N_{z^b=k|\wp^b=l} = \sum_i \mathbb{I}[z_i^b = k \wedge \wp_i^b = l]$ and similarly for $N_{\mathbf{x}^a|\wp^a}$. Given these counts, for each value of $k, l$, we now create the following vector: $\mathbf{v}_{kl} = \alpha \tau_k / (\alpha \tau_k + \mathbf{n}_{k|l} - 1)$ with $\mathbf{n}_{k|l} = [1, 2, .., N_{k|l}]$. We then draw a number $N_{k|l}$ Bernoulli random variables with probability of success given by the elements of $\mathbf{v}$, which we call[3] $\mathbf{s}_{k|l}^t$ and compute their sum across $t$: $S_{k|l} = \sum_t \mathbf{s}_{k|l}^t$. This procedure is equivalent to running a Chinese restaurant process (CRP) with $N_{k|l}$ customers and only keeping track of how many tables become occupied. We will denote it with $S_{k|l} \sim \mathcal{A}[N_{k|l}, \alpha \tau_k]$ after Antoniak [8]. Next we compute $S_k = \sum_l S_{k|l}$ and sample $\tau$ from a Dirichlet distribution, $\tau \sim \mathcal{D}[\gamma, S_1, .., S_k]$. Note that $\tau$ is a distribution over $K^a + 1$ states, where we now denote with $K^a$ the number of *occupied* states. If the state corresponding to $\gamma$ is picked, a new state is created and we increment $K^a \leftarrow K^a + 1$. If on the other hand a state becomes empty, we remove it from

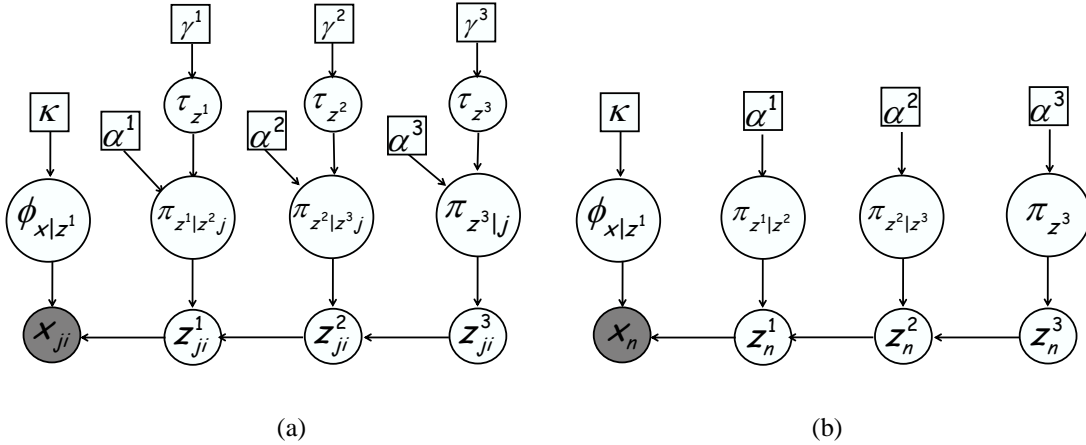

<div align="center">(a)　　　　　　　　　　　　　　(b)</div>

Figure 3: Graphical representation for (a) Pachinko Allocation and (b) Nested DP.

the list and we decrement $K^a \leftarrow K^a - 1$. This will allow assignment variables to add or remove states adaptively[4].

**Sampling $\mathbf{z}|(\boldsymbol{\tau}, \mathbf{x})$:** The conditional probability of all $\{\mathbf{z}_i, \mathbf{x}_i\}$ variables jointly (for fixed $i$) is given by,

$$P(\mathbf{x}_i, \mathbf{z}_i | \mathbf{z}_{\neg i}, \mathbf{x}_{\neg i}, \boldsymbol{\tau}, \boldsymbol{\alpha}) = \prod_a F(\mathbf{x}_i^a | \mathbf{x}_{\neg i}^a, \wp_{\neg i}^a) \prod_b \frac{\alpha^b \boldsymbol{\tau}_{z_i^b} + N_{z_i^b | \wp_i^b}^{\neg i}}{\alpha^b + N_{\wp_i^b}^{\neg i}} \tag{1}$$

where $N_{z_i^b | \wp_i^b}^{\neg i}$ is the number data-cases assigned to group $\mathbf{z}_i^b$ for variable $b$ and its parents assigned to group $\wp_i^b$, *where we exclude data-case $i$ from this count*. Also,

$$F(\mathbf{x}_i^a | \mathbf{x}_{\neg i}^a, \wp_{\neg i}^a = k) = \frac{\int d\boldsymbol{\phi}_k \, P(\mathbf{x}_i^a | \boldsymbol{\phi}_k) \prod_{i' \backslash i: \wp_{i'}^a = k} P(\mathbf{x}_{i'}^a | \boldsymbol{\phi}_k) \, P(\boldsymbol{\phi}_k)}{\int d\boldsymbol{\phi}_k \prod_{i' \backslash i: \wp_{i'}^a = k} P(\mathbf{x}_{i'}^a | \boldsymbol{\phi}_k) \, P(\boldsymbol{\phi}_k)} \tag{2}$$

Importantly, *equation 1 follows the structure of the original Bayes net*, where each term has the form of a conditional distribution $P(\mathbf{z}_i^a | \wp_i^a)$ and is based on sufficient statistics collected over all the other data-cases. Hence, we can use the structure of the Bayes net to sample the assignment variables jointly across the BN (for data-case $i$). The general technique that allows one to exploit network structure is 'forward-filtering-backward-sampling' (FFBS) [3]. Assume for instance that the network is a tree. In that case we first propagate information from the leaves to the root, computing the probabilities $P(\mathbf{z}^b | \{\mathbf{x}^{b\downarrow}\})$ as we go where '$\downarrow$' means that we compute a marginal conditioned on 'downstream' evidence. When we reach the root we draw a sample from $P(\mathbf{z}^{\text{root}} | \{\mathbf{x}^b\})$. Finally, we work our way back to the leaves, conditioning on drawn samples (which summarize upstream information) and using the marginal probabilities $P(\mathbf{z}^b | \{\mathbf{x}^{b\downarrow}\})$ cashed during the filtering phase to represent downstream evidence. For networks with higher treewidth we can extend this technique to junction trees. Alternatively, one can use cut-set sampling techniques [9].

## 5 ISBN for Structured Data

In section 2 we introduced an index $\mathfrak{n}$ to label the known structure of the data. The simplest nontrivial example is given by the HDP, where $\mathfrak{n} = (ji)$ indexing e.g. documents and words. In this case the CPT $\boldsymbol{\pi}_{z|j}$ is not shared across all data, but is specific to a document. Next consider Fig.1c where $\mathfrak{n} = (kji)$ is labelling for instance words $(i)$ in chapters $(j)$ in books $(k)$. The first level CPT $\boldsymbol{\pi}_{z|kj}$ is specific to chapters (and hence books) and is sampled from a Dirichlet distribution with mean given

by a second level CPT $\boldsymbol{\rho}_{z|k}$ specific to books, which in turn is sampled from a Dirichlet distribution with mean $\boldsymbol{\tau}_z$, which finally is sampled from a Dirichlet prior with parameters $\boldsymbol{\gamma}$. Sampling occurs again in two phases: sampling $\boldsymbol{\rho}, \boldsymbol{\tau}|\mathbf{x}, \mathbf{z}$ and $\mathbf{z}|\boldsymbol{\rho}, \boldsymbol{\tau}, \mathbf{x}$ while marginalizing out $\boldsymbol{\pi}, \boldsymbol{\phi}$.

To sample from $\boldsymbol{\rho}, \boldsymbol{\tau}$ we compute counts $N_{u|m,jk}$ which is the number of times words were assigned in chapter $j$ and book $k$ to the joint state $z = u, \wp = m$. We then work our way up the stack, sampling new count arrays $S, R$ as we go, and then down again sampling the CPTs $(\boldsymbol{\tau}, \boldsymbol{\rho})$ using these count arrays[5]. Note that this is just one step of Gibbs sampling from $P(\boldsymbol{\tau}, \boldsymbol{\rho}|\mathbf{z}, \mathbf{x})$ and does *not* (unlike the other phase for $\mathbf{z}$) generate an equilibrium sample from this conditional distribution.

$$\uparrow: \ s_{u|jkm} \sim \mathcal{A}[N_{u|jkm}, \alpha\boldsymbol{\rho}_{u|k}] \ \rightarrow \ S_{u|k} = \sum_{j,m} s_{u|jkm} \ \rightarrow \ r_{u|k} \sim \mathcal{A}[S_{u|k}, \beta\boldsymbol{\tau}_u] \ \rightarrow \ R_u = \sum_k t_{u|k}$$

$$\downarrow: \ \boldsymbol{\tau}_u \sim \mathcal{D}[(\gamma, R_u)] \ \rightarrow \ \boldsymbol{\rho}_{u|k} \sim \mathcal{D}[\beta\boldsymbol{\tau}_u + S_{u|k}] \tag{3}$$

A similar procedure is defined for the priors of $\phi$ and extensions to deeper stacks are straightforward.

If all $\mathbf{z}$ variables carry the same index $\mathfrak{n}$, sampling $\mathbf{z}_{\mathfrak{n}}$ given the hierarchical priors is very similar to the FFBS procedure described in the previous section, except that the count arrays may carry a subset of the indices from $\mathfrak{n}$, e.g. $N_{z|\wp jk}^{\neg ijk}$. Since these counts are specific to a chapter they are typically smaller resulting in a higher variance for the samples $\mathbf{z}$. If two neighboring $\mathbf{z}$ variables carry *different subsets* of $\mathfrak{n}$ labels, e.g. node $\mathbf{z}_j^0$ in Fig.2c, the conditional distributions are harder to compute. The general rule is to identify and remove all $\mathbf{z}'$ variables that are impacted by changing the value for $\mathbf{z}$ under consideration, e.g. $\{\mathbf{z}_{jw}^1, \ \forall w \cup \mathbf{z}_{jf}^2, \ \forall f\}$ in Fig.2c if we resample $\mathbf{z}_j^0$. To compute the conditional probability we set $\mathbf{z} = k$ and add the impacted variables $\mathbf{z}'$ back into the system, one-by-one in an arbitrary order and assigning them to their old values.

It is also instructive to place DP priors (instead of HDP priors) of the form $\mathcal{D}[\alpha^b/K^b]$ directly on $\boldsymbol{\pi}$ (skipping $\boldsymbol{\tau}$). In taking the infinite limit the conditional distribution for existing states $\mathbf{z}^b$ becomes directly proportional to $N_{z^b|\wp^b}$ (the $\alpha^b\boldsymbol{\tau}_{z^b}$ term is missing). This has the effect that a new state $z^b = k$ that was discovered for some parent state $\wp^b = l$ will not be available to other parent states, simply because $N_{k|l'} = 0$, $l' \neq l$. The result is that the state space forks into a tree structure as we move down the Bayes net. When the network structure is a linear chain, this model is equivalent to the 'nested-DP' introduced in [10] as a prior on tree-structures. The corresponding Bayes net is depicted in Fig.3b. A chain of length 1 is of course just a Dirichlet process mixture model. A DP prior is certainly appropriate for nodes $\mathbf{z}^b$ with CPTs that do not depend on other parents or additional labels, e.g. nodes $\mathbf{z}^3$ and $\mathbf{z}^4$ in Fig.1a. Interestingly, an HDP would also be possible and would result in a different model. We will however follow the convention that we will use the minimum depth necessary for modelling the structure of the data.

## 6   Examples

**Example: HDP**   Perhaps the simplest example is an HDP itself, see Fig.1b. It consists of a single topic node and a single observation node. If we make $\phi$ depend on the item index $i$, i.e. $\phi_{x|z,i}$, we obtain the infinite version of the 'user rating profile' (URP) model [11]. If we make $\phi$ depend on $j$ instead and add a prior: $\psi_{x|z} \rightarrow \phi_{x|z,j}$, we obtain an "HDP with random effects" [12] which has the benefit that shared topics across documents can vary slightly relative to each other.

**Example: Infinite State Chains**   The 'Pachinko allocation model' (PAM) [13] consists of a linear chain of assignment variables with document specific transition probabilities, see Fig.3a. It was proposed to model correlations between topics. The infinite version of this is clearly an example of an ISBN. An equivalent Chinese restaurant process formulation was published in [14]. A slight variation on this architecture was described in [15] (POM). Here, images are modeled as mixtures over parts and parts were modeled as mixtures over visual words. Finally, a visual word is a distribution over features. POM is only subtly different from PAM (see Fig.3a) in that parts are not image-specific distributions over words, and so the distribution $\pi_{z^1|z^2}$ does not depend on $j$.

**Example: BiHDP**   This model, depicted in Fig.2a has a data variable $\mathbf{x}_{ji}$ and two parent topic variables $\mathbf{z}_{ji}^1$ and $\mathbf{z}_{ji}^2$. One can think of $j$ as the customer index and $i$ as the product index (and no IID repeated index). The value of $\mathbf{x}$ is the rating of that customer for that product. The hidden variables

$\mathbf{z}_{ji}^1$ and $\mathbf{z}_{ji}^2$ represent product groups and customer groups. Every data entry is assigned to both a customer group and a product group which together determine the factor from which we sample the rating. Note that the difference between the assignment variables is that their corresponding CPTs $\boldsymbol{\pi}_{z^1,j}$ and $\boldsymbol{\pi}_{z^2,i}$ depend on $j$ and $i$ respectively. Extensions are easily conceived. For instance, instead of two modalities, we can model multiple modalities (e.g. customers, products, year). Also, single topics can be generalized to hierarchies of topics, so every branch becomes a PAM. Note that for unobserved $\mathbf{x}_{ji}$ values (not all products have been rated by all customers) the corresponding $\mathbf{z}_{ji}^a, \mathbf{z}_{ji}^b$ are "dangling" and can be integrated out. The result is that we should skip that variable in the Gibbs sampler.

**Example: The Mixed-Membership Stochastic Block Model[16]** This model is depicted in Fig.2b. The main difference with HDP is that (like BiHDP) $\boldsymbol{\pi}$ depends on two parent states $\mathbf{z}_{i \to j}$ and $\mathbf{z}_{j \to i}$ by which we mean that item $i$ has chosen topic $\mathbf{z}_{i \to j}$ to interact with item $j$ and vice versa. However, (unlike BiHDP) those topic states share a common distribution $\boldsymbol{\pi}$. Indices only run over distinct pairs $i > j$. These features make the model suitable for modeling social interaction networks or protein-protein interaction networks. The hidden variables jointly label the type of interaction that was used to generate 'matrix-element' $\mathbf{x}_{ij}$.

**Example: The Multimedia Model** In the above examples we had a single observed variable in the graphical model (repeated over $ij$). The model depicted in Fig.2c has two observed variables and an assignment variable that is not repeated over items. We can think of the middle node $\mathbf{z}_j^0$ as the class label for a web-page $j$. The left branch can then model words on the web-page while the right branch can model visual features on the web-page. Since no sharing is required for $\mathbf{z}_j^0$ we used a Dirichlet prior. The other variables have the usual HDP priors.

# 7  Experiments

To illustrate the ideas we implemented two models: BiHDP of Fig.2a and the "probabilistic object model" (POM), explained in the previous section.

**Market Basket Data** In this experiment we investigate the performance of BiHDP on a synthetic market basket dataset. We used the IBM Almaden association and sequential patterns generator to create this dataset [17]. This is a standard synthetic transaction dataset generator often used by the association research community. The generated data consists of purchases from simulated groups of customers who have similar buying habits. Similarity of buying habits refers to the fact that customers within a group buy similar groups of items. For example, items like strawberries and cream are likely to be in the same item group and thus are likely to be purchased together in the same market basket. The following parameters were used to generate data for our experiments: 1M transactions, 10K customers, 1K different items, 4 items per transaction on average, 4 item groups per customer group on average, 50 market basket patterns, 50 customer patterns. Default values were used for the remaining parameters.

The two assignment variables correspond to customers and items respectively. For a given pair of customer and item groups, a binomial distribution was used to model the probability of a customer group making a purchase from that item group. A collapsed Gibbs sampler was used to fit the model. After 1000 epochs the system converged to 278 customer groups and 39 item factors. Fig.4 shows the results. As can be seen, most item groups correspond directly to the hidden ground truth data. The conclusion is that the model can learn successfully the hidden structure in the data.

**Learning Visual Vocabularies** LDA has also gained popularity to model images as collections of features. The visual vocabulary is usually determined in a preprocessing step where k-means is run to cluster features collected from the training set. In [15] a different approach was proposed in which the visual word vocabulary was learned jointly with fitting the parameters of the model. This can have the benefit that the vocabulary is better adapted to suit the needs of the model. Our extension of their PLSA-based model is the infinite state model given by Fig.3a with 2 hidden variables (instead of 3) and $\boldsymbol{\pi}_{z^1|z^2}$ independent of $j$. $\mathbf{x}$ is modeled as a Gaussian-distributed random variable over feature values, $\mathbf{z}^1$ represents the word identity and $\mathbf{z}^2$ is the topic index.

We used the Harris interest-point detector and $21 \times 21$ patches centered on each interest point as input to the algorithm. We normalized the patches to have zero mean. Next we reduced the dimensionality of detections from 441 to 100 using PCA. The procedure described above generates a set of between

| | |
|---|---|
| Learned: 223, **619**, 271, 448, 39, **390** | True: 223, 271, 448, 39, 427, 677 |
| Learned: 364, **250**, 718, 952, 326, **802** | True: 364, 718, 952, 326, 542, 98 |
| Learned: 159, 563, 780, 995, 103, 216, **598**, 72 | True: 159, 563, 780, 995, 103, 216, 542, 72 |
| Learned: 227, 130, 862, 991, 904, 213 | True: 227, 130, 862, 991, 904, 213 |
| Learned: 953, 175, 956, 385, 269, 14, **64** | True: 953, 175, 956, 385, 269, 14, 956 |
| Learned: 49, 657, 906, 604, 229 | True: 49, 657, 906, 604, 229 |
| Learned: 295, 129, 662, 922, 705, **210** | True: 295, 129, 662, 922, 705, 68 |
| Learned: 886, 460, 471, 933, **544** | True: 886, 460, 471, 933, 917 |
| Learned: 489, 818, 927, 378, 64, **710** | True: 489, 818, 927, 378, 64, 247 |
| Learned: 776, 224, 139, 379 | True: 776, 224, 139, 379 |

Figure 4: The 10 most popular item groups learned by the BiHDP model (left) compared to ground truth item groups for market basket data (right). Learned items are ordered by decreasing popularity. Ground truth items have no associated weight; therefore, they were ordered to facilitate comparison with the left row. Non-matching items are shown in boldface.

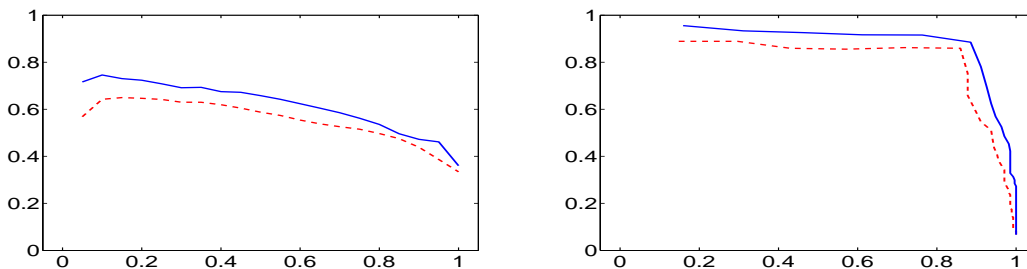

Figure 5: Precision Recall curves for Caltech-4 dataset (left) and turntable dataset (right). Solid curve represents POM and dashed curve represents LDA.

50 and 400, 100-dimensional detections per image. Experiments were performed using the Caltech-4 and 'turntable' datasets. For Caltech-4 we used 130 randomly sampled images from each of the 4 categories for training. LDA was fit using 500 visual words and 50 parts (which we found to give the best results). The turntable database contains images of 15 toy objects. The objects were placed on a turntable and photographed every 5 degrees. We have used angles 15, 20, 25, 35, 40, and 45 for training, and angles 10, 30, and 50 for testing. LDA used 15 topics and 200 visual words (which again was optimal). LDA was then fitted to both datasets using Gibbs sampling. We initialized POM with the output of LDA to make sure the comparison involved similar modes of the distribution.

The precision-recall curves for this dataset are shown in Fig.5. Images were labelled by choosing the majority class across the 11 most similar retrieved images. Similarity was measured as the probability of the query image given the part probabilities of the retrieved image.

These experiments show that ISBNs can be successfully implemented. We are not interested in claiming superiority of ISBNs, but rather hope to convey that ISBNs are a convenient tool to design models and to facilitate the search for the number of latent states.

## 8   Discussion

We have presented a unified framework to organize the fast growing class of 'topic models'. By merging ideas from Bayes nets, nonparametric Bayesian statistics and topic models we have arrived at a convenient framework to 1) extend existing models to infinite state spaces, 2) reason about and design new models and 3) derive efficient inference algorithms that exploit the structure of the underlying Bayes net.

Not every topic model naturally fits the suit of an ISBN. For instance, the infinite HMM [18] is like a POM model with emission states, but with a single transition probability shared across time. When marginalizing out $\pi$ this has the effect of coupling all $z$ variables. An efficient sampler for this model was introduced in [19]. Also, in [10, 20] models were studied where a word can be emitted at

any node corresponding to a topic variable **z**. We would need an extra switching variable to fit this into the ISBN framework.

We are currently working towards a graphical interface where one can design ISBN models by attaching together $H_kDP$ modules and where the system will automatically perform the inference necessary for the task at hand.

**Acknowledgements**

This material is based upon work supported by the National Science Foundation under Grant No. 0447903 and No. 0535278 and by ONR under Grant No. 00014-06-1-0734.

## Footnotes

[1]We will often avoid writing the super-indices $a, b$ when it is clear from the context, e.g. $\phi^a_{x^a|\wp^a} = \phi_{x^a|\wp^a}$.

[2]Note that these can be also used to compute Rao-Blackwellised estimates of $\pi$ a $\phi$, i.e. $\mathbb{E}[\pi_{z^b|\wp^b}] = (\alpha^b \tau_z^b + N_{z^b|\wp^b})/(\alpha^b + N_{\wp^b})$ and similarly for $\phi$.

[3]These variables are so called *auxiliary* variables to facilitate sampling $\tau$.

[4]We adopted the name 'infinite state Bayesian network' because the $(K^a + 1)^{\text{th}}$ state actually represents an infinite pool of indistinguishable states.

[5]Teh's code npbayes-r21, (available from his web-site) does in fact implement this sampling process.

# References

[1] D. M. Blei, A. Y. Ng, and M. I. Jordan. Latent Dirichlet allocation. *Journal of Machine Learning Research*, 3:993–1022, 2003.

[2] Y. W. Teh, M. I. Jordan, M. J. Beal, and D. M. Blei. Hierarchical Dirichlet processes. *To appear in Journal of the American Statistical Association*, 2006.

[3] S. L. Scott. Bayesian methods for hidden Markov models, recursive computing in the 21st century. volume 97, pages 337–351, 2002.

[4] T. Minka. Estimating a dirichlet distribution. Technical report, 2000.

[5] M.D. Escobar and M. West. Bayesian density estimation and inference using mixtures. *Journal of the American Statistical Association*, 90:577–588, 1995.

[6] T.L. Griffiths and M. Steyvers. A probabilistic approach to semantic representation. In *Proceedings of the 24th Annual Conference of the Cognitive Science Society*, 2002.

[7] Y.W. Teh, D. Newman, and M. Welling. A collapsed variational bayesian inference algorithm for latent dirichlet allocation. In *NIPS*, volume 19, 2006.

[8] C.E. Antoniak. Mixtures of Dirichlet processes with applications to bayesian nonparametric problems. *The Annals of Statistics*, 2:1152–1174, 1974.

[9] B. Bidyuk and R. Dechter. Cycle-cutset sampling for Bayesian networks. In *Sixteenth Canadian Conf. on AI*, 2003.

[10] David Blei, Thomas L. Griffiths, Michael I. Jordan, and Joshua B. Tenenbaum. Hierarchical topic models and the nested chinese restaurant process. In *Neural Information Processing Systems 16*, 2004.

[11] B. Marlin. Modeling user rating profiles for collaborative filtering. In *Advances in Neural Information Processing Systems 16*. 2004.

[12] S. Kim and P. Smyth. Hierarchical dirichlet processes with random effects. In *NIPS*, volume 19, 2006.

[13] W. Li and A. McCallum. Pachinko allocation: Dag-structured mixture models of topic correlations. In *Proceedings of the 23rd international conference on Machine learning*, pages 577–584, 2006.

[14] W. Li, A. McCallum, and D. Blei. Nonparametric bayes pachinko allocation. In *UAI*, 2007.

[15] D. Larlus and F. Jurie. Latent mixture vocabularies for object categorization. In *British Machine Vision Conference*, 2006.

[16] E. Airoldi, D. Blei, E. Xing, and S. Fienberg. A latent mixed membership model for relational data. In *LinkKDD '05: Proceedings of the 3rd international workshop on Link discovery*, pages 82–89, 2005.

[17] R. Agrawal, T. Imielinski, and A. Swami. Mining associations between sets of items in massive databases. In *Proc. of the ACM-SIGMOD 1993 Intl Conf on Management of Data*, 1993.

[18] M.J. Beal, Z. Ghahramani, and C.E. Rasmussen. The infinite hidden markov model. In *NIPS*, pages 577–584, 2001.

[19] Y. W. Teh, D. Görür, and Z. Ghahramani. Stick-breaking construction for the Indian buffet process. In *Proceedings of the International Conference on Artificial Intelligence and Statistics*, volume 11, 2007.

[20] W. Li D. Mimno and A. McCallum. Mixtures of hierarchical topics with pachinko allocation. In *Proceedings of the 21st International Conference on Machine Learning*, 2007.

